# Local Supervised Learning through Space Partitioning

**Joseph Wang**
Dept. of Electrical and Computer Engineering
Boston University
Boston, MA 02116
joewang@bu.edu

**Venkatesh Saligrama**
Dept. of Electrical and Computer Engineering
Boston University
Boston, MA 02116
srv@bu.edu

## Abstract

We develop a novel approach for supervised learning based on adaptively partitioning the feature space into different regions and learning local region-specific classifiers. We formulate an empirical risk minimization problem that incorporates both partitioning and classification in to a single global objective. We show that space partitioning can be equivalently reformulated as a supervised learning problem and consequently any discriminative learning method can be utilized in conjunction with our approach. Nevertheless, we consider locally linear schemes by learning linear partitions and linear region classifiers. Locally linear schemes can not only approximate complex decision boundaries and ensure low training error but also provide tight control on over-fitting and generalization error. We train locally linear classifiers by using LDA, logistic regression and perceptrons, and so our scheme is scalable to large data sizes and high-dimensions. We present experimental results demonstrating improved performance over state of the art classification techniques on benchmark datasets. We also show improved robustness to label noise.

## 1   Introduction

We develop a novel approach for supervised learning based on adaptively partitioning the feature space into different regions and learning local region classifiers. Fig. 1 (left) presents one possible architecture of our scheme (others are also possible). Here each example passes through a cascade of reject classifiers ($g_j$'s). Each reject classifier, $g_j$, makes a binary decision and the observation is either classified by the associated region classifier, $f_j$, or passed to the next reject classifier. Each reject classifier, $g_j$, thus partitions the feature space into regions. The region classifier $f_j$ operates only on examples within the local region that is consistent with the reject classifier partitions.

We incorporate both feature space partitioning (reject classifiers) and region-specific classifiers into a single global empirical risk/loss function. We then optimize this global objective by means of coordinate descent, namely, by optimizing over one classifier at a time. In this context we show that each step of the coordinate descent can be reformulated as a supervised learning problem that seeks to optimize a 0/1 empirical loss function. This result is somewhat surprising in the context of partitioning and has broader implications. First, we can now solve feature space partitioning through empirical risk function minimization(ERM) and so powerful existing methods including boosting, decision trees and kernel methods can be used in conjunction for training flexible partitioning classifiers.

Second, because data is usually locally "well-behaved," simpler region-classifiers, such as linear classifiers, often suffice for controlling local empirical error. Furthermore, since complex boundaries for partitions can be approximated by piecewise linear functions, feature spaces can be partitioned to arbitrary degree of precision using linear boundaries (reject classifiers). Thus the combination of piecewise linear partitions along with linear region classifiers has the ability to adapt to complex data sets leading to low training error. Yet we can prevent overfitting/overtraining by optimizing the

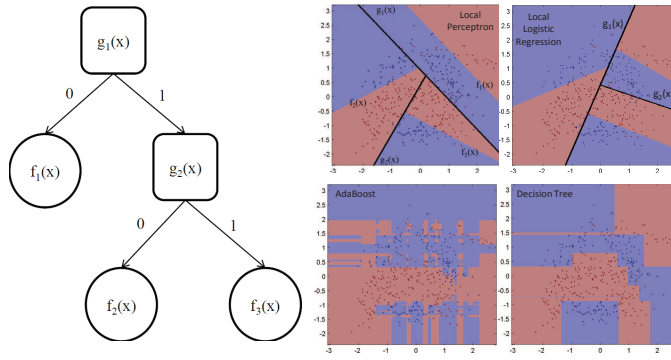

Figure 1: **Left:** Architecture of our system. Reject Classifiers, $g_j(x)$, partition space and region classifiers, $f_j(x)$, are applied locally within the partitioned region. **Right:** Comparison of our approach (upper panel) against Adaboost and Decision tree (lower panel) on the banana dataset[1]. We use linear perceptrons and logistic regression for training partitioning classifier and region classifiers. Our scheme splits with 3 regions and does not overtrain unlike Adaboost.

number of linear partitions and linear region classifiers, since the VC dimension of such a structure is reasonably small. In addition this also ensures significant robustness to label noise. Fig. 1 (right) demonstrates the substantial benefits of our approach on the banana dataset[1] over competing methods such as boosting and decision trees, both of which evidently overtrain.

Limiting reject and region classifiers to linear methods has computational advantages as well. Since the datasets are locally well-behaved we can locally train with linear discriminant analysis (LDA), logistic regression and variants of perceptrons. These methods are computationally efficient in that they scale linearly with data size and data dimension. So we can train on large high-dimensional datasets with possible applications to online scenarios.

Our approach naturally applies to multi-class datasets. Indeed, we present some evidence that shows that the partitioning step can adaptively cluster the dataset into groups and letting region classifiers to operate on simpler problems. Additionally linear methods such as LDA, Logistic regression, and perceptron naturally extend to multi-class problems leading to computationally efficient and statistically meaningful results as evidenced on challenging datasets with performance improvements over state of the art techniques.

## 1.1 Related Work

Our approach fits within the general framework of combining simple classifiers for learning complex structures. Boosting algorithms [2] learn complex decision boundaries characterized as a weighted linear combination of weak classifiers. In contrast our method takes unions and intersections of simpler decision regions to learn more complex decision boundaries. In this context our approach is closely related to decision trees. Decision trees are built by greedily partitioning the feature space [3]. One main difference is that decision trees typically attempt to greedily minimize some loss or a heuristic, such as region purity or entropy, at each split/partition of the feature space. In contrast our method attempts to minimize global classification loss. Also decision trees typically split/partition a single feature/component resulting in unions of rectangularly shaped decision regions; in contrast we allow arbitrary partitions leading to complex decision regions.

Our work is loosely related to so called coding techniques that have been used in multi-class classification [4, 5]. In these methods a multiclass problem is decomposed into several binary problems using a code matrix and the predicted outcomes of these binary problems are fused to obtain multiclass labels. Jointly optimizing for the code matrix and binary classification is known to be NP hard [6] and iterative techniques have been proposed [7, 8]. There is some evidence (see Sec. 3) that suggests that our space partitioning classifier groups/clusters multiple classes into different regions; nevertheless our formulation is different in that we do not explicitly code classes into different regions and our method does not require fusion of intermediate outcomes.

Despite all these similarities, at a fundamental level, our work can also be thought of as a somewhat complementary method to existing supervised learning algorithms. This is because we show that space partitioning itself can be re-formulated as a supervised learning problem. Consequently, any

existing method, including boosting and decision trees, could be used as a method of choice for learning space partitioning and region-specific decision functions.

We use simple linear classifiers for partitioning and region-classifiers in many of our experiments. Using piecewise combinations of simple functions to model a complex global boundary is a well studied problem. Mixture Discriminant Analysis (MDA), proposed by Hastie *et al.* [9], models each class as a mixture of gaussians, with linear discriminant analysis used to build classifiers between estimated gaussian distributions. MDA relies upon the structure of the data, assuming that the true distribution is well approximated by a mixture of Gaussians. Local Linear Discriminant Analysis (LLDA) , proposed by Kim *et al.* [10], clusters the data and performs LDA within each cluster. Both of these approaches partition the data then attempt to classify locally. Partitioning of the data is independent of the performance of the local classifiers, and instead based upon the spatial structure of the data. In contrast, our proposed approach partitions the data based on the performance of classifiers in each region. A recently proposed alternative approach is to build a global classifier ignoring clusters of errors, and building separate classifiers in each error cluster region [11]. This proposed approach greedily approximates a piecewise linear classifier in this manner, however fails to take into account the performance of the classifiers in the error cluster regions. While piecewise linear techniques have been proposed in the past [12, 13], we are unaware of techniques that learn piecewise linear classifiers based on minimizing global ERM and allows any discriminative approach to be used for partitioning and local classification, and also extends to multiclass learning problems.

## 2 Learning Space Partitioning Classifiers

The goal of supervised classification is to learn a function, $f(x)$, that maps features, $x \in \mathcal{X}$, to a discrete label, $y \in \{1, 2, \ldots, c\}$, based on training data, $(x_i, y_i)$, $i = 1, 2, \ldots, n$. The empirical risk/loss of classifier $f$ is:

$$R(f) = \frac{1}{n} \sum_{i=1}^{n} \mathbb{1}_{\{f(x_i) \neq y_i\}}$$

Our goal is empirical risk minimization(ERM), namely, to minimize $R(f)$ over all classifiers, $f(\cdot)$, belonging to some class $\mathcal{F}$. It is well known that the complexity of the family $\mathcal{F}$ dictates generalization errors. If $\mathcal{F}$ is too simple, it often leads to large bias errors; if the family $\mathcal{F}$ is too rich, it often leads to large variance errors. With this perspective we consider a family of classifiers (see Fig. 1 that adaptively partitions data into regions and fits simple classifiers within each region. We predict the output for a test sample, $x$, based on the output of the trained simple classifier associated with the region $x$ belongs to. The complexity of our family of classifiers depends on the number of local regions, the complexity of the simple classifiers in each region, and the complexity of the partitioning. In the sequel we formulate space partitioning and region-classification into a single objective and show that space partitioning is equivalent to solving a binary classification problem with 0/1 empirical loss.

### 2.1 Binary Space Partitioning as Supervised Learning

In this section we consider learning binary space partitioning for ease of exposition. The function $g(\cdot)$ partitions the space by mapping features, $x \in \mathcal{X}$, to a binary label, $z \in \{0, 1\}$. Region classifiers $f_0(x)$, $f_1(x)$ operate on the respective regions generated by $g(x)$ (see Fig. 1). The empirical risk/loss associated with the binary space partitioned classifiers is given by:

$$R(g, f_0, f_1) = \frac{1}{n} \sum_{i=1}^{n} \mathbb{1}_{\{g(x_i)=0\}} \mathbb{1}_{\{f_0(x_i) \neq y_i\}} + \frac{1}{n} \sum_{i=1}^{n} \mathbb{1}_{\{g(x_i)=1\}} \mathbb{1}_{\{f_1(x_i) \neq y_i\}} \quad (1)$$

Our goal is to minimize the empirical error jointly over the family of functions $g(\cdot) \in \mathcal{G}$ and $f_i(\cdot) \in \mathcal{F}$. From the above equation, when the partitioning function $g(\cdot)$ is fixed, it is clear how one can view choice of classifiers $f_0(\cdot)$ and $f_1(\cdot)$ as ERM problems. In contrast, even when $f_0$, $f_1$ are fixed, it is unclear how to view minimization over $g \in \mathcal{G}$ as an ERM. To this end let, $\ell_0^{(i)}$, $\ell_1^{(i)}$ indicate whether or not classifier $f_0$, $f_1$ makes an error on example $x^{(i)}$ and let $S$ denote the set of instances where the classifier $f_0$ makes errors, namely,

$$\ell_i^0 = \mathbb{1}_{\{f_0(x_i) \neq y_i\}}, \; \ell_i^1 = \mathbb{1}_{\{f_1(x_i) \neq y_i\}}, \; S = \{i \mid \ell_i^0 = 1\} \quad (2)$$

We can then rewrite Eq. 1 as follows:

$$
\begin{aligned}
R(g, f_0, f_1) &= \frac{1}{n}\sum_{i=1}^{n}\ell_i^0 \mathbb{1}_{\{g(x_i)=0\}} + \frac{1}{n}\sum_{i=1}^{n}\ell_i^1 \mathbb{1}_{\{g(x_i)=1\}} \\
&= \frac{1}{n}\sum_{i\in S}\mathbb{1}_{\{g(x_i)=0\}} + \frac{1}{n}\sum_{i\in S}\ell_i^1 \mathbb{1}_{\{g(x_i)=1\}} + \frac{1}{n}\sum_{i\notin S}\ell_i^1 \mathbb{1}_{\{g(x_i)=1\}} \\
&= \frac{1}{n}\sum_{i\in S}\mathbb{1}_{\{g(x_i)=0\}} + \frac{1}{n}\sum_{i\in S}\ell_i^1 (1 - \mathbb{1}_{\{g(x_i)=0\}}) + \frac{1}{n}\sum_{i\notin S}\ell_i^1 \mathbb{1}_{\{g(x_i)=1\}} \\
&= \frac{1}{n}\sum_{i\in S}(1-\ell_i^1)\mathbb{1}_{\{g(x_i)=0\}} + \underbrace{\frac{1}{n}\sum_{i\in S}\ell_i^1}_{\text{indep. of } g} + \frac{1}{n}\sum_{i\notin S}\ell_i^1 \mathbb{1}_{\{g(x_i)=1\}}
\end{aligned}
$$

Note that for optimizing $g \in \mathcal{G}$ for fixed $f_0$, $f_1$, the second term above is constant. Furthermore, by consequence of Eq. 2 we see that the first and third terms can be further simplified as follows:

$$
\frac{1}{n}\sum_{i\in S}(1-\ell_i^1)\mathbb{1}_{\{g(x_i)=0\}} = \frac{1}{n}\sum_{i\in S}(1-\ell_i^1)\mathbb{1}_{\{g(x_i)\neq \ell_i^0\}}; \quad \frac{1}{n}\sum_{i\notin S}\ell_i^1 \mathbb{1}_{\{g(x_i)=1\}} = \frac{1}{n}\sum_{i\notin S}\ell_i^1 \mathbb{1}_{\{g(x_i)\neq \ell_i^0\}}
$$

Putting all this together we have the following lemma:

**Lemma 2.1.** *For a fixed $f_0$, $f_1$ the problem of choosing the best binary space partitions, $g(\cdot)$ in Eq. 1 is equivalent to choosing a binary classifier $g$ that optimizes following 0/1 (since $w_i \in \{0, 1\}$) empirical loss function:*

$$
\tilde{R}(g) = \frac{1}{n}\sum_{i=1}^{n} w_i \mathbb{1}_{\{g(x_i)\neq \ell_i^0\}}, \quad \text{where } w_i = \left\{ \begin{array}{l} 1,\ \ell_i^0 \neq \ell_i^1 \\ 0,\ \text{otherwise} \end{array} \right.
$$

The composite classifier $F(x)$ based on the reject and region classifiers can be written compactly as $F(x) = f_{g(x)}(x)$. We observe several aspects of our proposed scheme:

**(1)** Binary partitioning is a binary classification problem on the training set, $(x_i, \ell_i^0)$, $i = 1, 2, \ldots, n$.

**(2)** The 0/1 weight, $w_i = 1$, is non-zero if and only if the classifiers disagree on $x_i$, i.e., $f_0(x_i) \neq f_1(x_i)$.

**(3)** The partitioning error is zero on a training example $x_i$ with weight $w_i = 1$ if we choose $g(x_i) = 0$ on examples where $f_0(x_i) = y_i$. In contrast if $f_0(x_i) \neq y_i$ the partitioning error can be reduced by choosing $g(x_i) = 1$, and thus **rejecting** the example from consideration by $f_0$.

## 2.2 Surrogate Loss Functions, Algorithms and Convergence

An important implication of Lemma 2.1 is that we can now use powerful learning techniques such as decision trees, boosting and SVMs for learning space partitioning classifiers. Our method is a coordinate descent scheme which optimizes over a single variable at a time. Each step is an ERM and so any learning method can be used at each step.

**Convergence Issues:** It is well known that that indicator losses are hard to minimize, even when the class of classifiers, $\mathcal{F}$, is nicely parameterized. Many schemes are based on minimizing surrogate losses. These surrogate losses are upper bounds for indicator losses and usually attempt to obtain large margins. Our coordinate descent scheme in this context is equivalent to describing surrogates for each step and minimizing these surrogates. This means that our scheme may not converge, let alone converge to a global minima, even when surrogates at each step are nice and convex. This is because even though each surrogate upper bounds indicator loss functions at each step, when put together they do not upper bound the global objective of Eq. 1. Consequently, we need a global surrogate to ensure that the solution does converge. Loss functions are most conveniently thought of in terms of margins. For notational convenience, in this section we will consider the case where the partition classifier, $g$, maps to labels $\ell \in \{-1, 1\}$, where a label of $-1$ and $1$ indicates classification by $f_0$ and $f_1$, respectively. We seek functions $\phi(z)$ that satisfy $\mathbb{1}_{z\leq 0} \leq \phi(z)$. Many such surrogates can be constructed using sigmoids, exponentials etc. Consider the classification function $g(x) = sign\,(h(x) > 0)$. The empirical error can be upper

bounded: $\mathbb{1}_{\ell g(x)=1} = \mathbb{1}_{-\ell h(x)\leq 0} \leq \phi(-\ell h(x))$ We then form a global surrogate for the empirical loss function. Approximating the indicator functions of the empirical risk/loss in Eq. 1 with surrogate functions, the global surrogate is given by:

$$\hat{R}(g, f_0, f_1) = \frac{1}{n}\sum_{i=1}^{n} \phi\left(h(x_i)\right)\phi\left(y_i f_0(x_i)\right) + \frac{1}{n}\sum_{i=1}^{n} \phi\left(-h(x_i)\right)\phi\left(y_i f_1(x_i)\right), \qquad (3)$$

which is an upper bound on Eq. 1. Optimizing the partitioning function $g(\cdot)$ can be posed as a supervised learning problem, resulting in the following lemma (see Supplementary for a proof):

**Lemma 2.2.** *For a fixed $f_0$, $f_1$ the problem of choosing the best binary space partitions, $g(\cdot)$ in Eq. 3 is equivalent to choosing a binary classifier $h$ that optimizes a surrogate function $\phi(\cdot)$:*

$$\hat{R}(g) = \frac{1}{2n}\sum_{i=1}^{2n} w_i \phi\left(h(x_i)r_i\right), \ r_i = \left\{ \begin{array}{ll} 1, & i < n+1 \\ -1, & otherwise \end{array} \right. , w_i = \left\{ \begin{array}{ll} \phi(f_0(x_i)y_i), & i < n+1 \\ \phi(f_1(x_i)y_i), & otherwise \end{array} \right. .$$

**Theorem 2.3.** *For any continuous surrogate $\phi(\cdot,\cdot)$, performing alternating minimization on the classifiers $f_0$, $f_1$, and $g$ converges to a local minima of Eq. 3, with a loss upper-bounding the empirical loss defined by Eq. 1.*

*Proof.* This follows directly, as this is coordinate descent on a smooth cost function. □

## 2.3 Multi-Region Partitioning

Lemma 2.1 can be used to also reduce multi-region space partitioning to supervised learning. We can obtain this reduction in one of several ways. One approach is to use pairwise comparisons, training classifiers to decide between pairs of regions. Unfortunately, the number of different reject classifiers scales quadratically, so we instead employ a greedy partitioning scheme using a cascade classifier.

Fig 1 illustrates a recursively learnt three region space partitioning classifier. In general the regions are defined by a cascade of binary reject classifiers, $g_k(x), k \in \{1, 2, \ldots, r-1\}$, where $r$ is the number of classification regions. Region classifiers, $f_k(x), k \in \{1, 2, \ldots, r\}$, map observations in the associated region to labels. At stage $k$, if $g_k(x) = 0$, an observation is classified by the region classifier, $f_k(x)$, otherwise the observation is passed to the next stage of the cascade. At the last reject classifier in the cascade, if $g_{r-1}(x) = 1$, the observation is passed to the final region classifier, $f_r(x)$. This ensures that only $r$ reject classifiers have to be trained for $r$ regions.

Now define for an arbitrary instance $(x, y)$ and fixed $\{g_j\}$, $\{f_j\}$, the 0/1 loss function at each stage $k$,

$$L_k(x, y) = \left\{ \begin{array}{ll} \left(\mathbb{1}_{\{g_k(x)=0\}}\right) \mathbb{1}_{\{f_k(x)\neq y\}} + \left(\mathbb{1}_{\{g_k(x)=1\}}\right) L_{k+1}(x, y) & \text{if } k < r \\ \mathbb{1}_{\{f_{r+1}(x)\neq y\}} & \text{if } k = r \end{array} \right. , \qquad (4)$$

We observe that $L_k(x, y) \in \{0, 1\}$ and is equal to zero if the example is classified correctly at current or future stages and one otherwise. Consequently, the aggregate 0/1 empirical risk/loss is the average loss over all training points at stage 1, namely,

$$R\left(g_1, g_2, \ldots, g_{r-1}, f_1, f_2, \ldots, f_r\right) = \frac{1}{n}\sum_{i=1}^{n} L_1(x_i, y_i) \qquad (5)$$

In the expression above we have made the dependence on reject classifiers and region-classifiers explicit. We minimize Eq. 5 over all $g_j$, $f_j$ by means of coordinate descent, namely, to optimize $g_k$ we hold $f_j$, $\forall j$ and $g_j$, $j \neq k$ fixed. Based on the expressions derived above the coordinate descent steps for $g_k$ and $f_k$ reduces respectively to:

$$g_k(\cdot) = \operatorname*{argmin}_{g \in \mathcal{G}} \frac{1}{n}\sum_{i=1}^{n} C_k(x_i)L_k(x_i, y_i), \ f_k(\cdot) = \operatorname*{argmin}_{f \in \mathcal{F}} \frac{1}{n}\sum_{i=1}^{n} C_k(x_i)\mathbb{1}_{\{f_k(x_i)\neq y_i\} \bigwedge \{g_k(x_i)=0\}}$$

$$(6)$$

where, $C_j(x) = \mathbb{1}_{\{\bigwedge_{i=1}^{j-1}\{g_i(x)=1\}\}}$, denotes whether or not an example makes it to the jth stage. The optimization problem for $f_k(\cdot)$ is exactly the standard 0/1 empirical loss minimization over training

**Algorithm 1** Space Partitioning Classifier

---

**Input:** Training data, $\{(x_i, y_i)\}_{i=1}^n$, number of classification regions, $r$
**Output:** Composite classifier, $F(\cdot)$
**Initialize:** Assign points randomly to $r$ regions
**while** F not converged **do**
  **for** $j = 1, 2, \ldots, r$ **do**
    Train region classifier $f_j(x)$ to optimize 0/1 empirical loss of Eq. (6).
  **end for**
  **for** $k = r - 1, r - 2, \ldots, 2, 1$ **do**
    Train reject classifier $g_k(x)$ to optimize 0/1 empirical loss of Eq. (7).
  **end for**
**end while**

---

data that survived upto stage $k$. On the other hand, the optimization problem for $g_k$ is exactly in the form where Lemma 2.1 applies. Consequently, we can also reduce this problem to a supervised learning problem:

$$g_k(\cdot) = \operatorname*{argmin}_{g \in \mathcal{G}} \frac{1}{n} \sum_{i=1}^n w_i \mathbb{1}_{\{g(x_i) \neq \ell_i\}}, \tag{7}$$

where

$$\ell_i = \begin{cases} 0 & \text{if } f_k(x_i) = y_i \\ 1 & \text{if } f_k(x_i) \neq y_i \end{cases} \quad \text{and} \quad w_i = \begin{cases} 1, & \ell_i \neq L_{k+1}(x_i, y_i),\ C_k(x) \neq 0 \\ 0, & \text{otherwise} \end{cases}.$$

The composite classifier $F(x)$ based on the reject and region classifiers can be written compactly as follows:

$$F(x) = f_s(x), \ s = \min\{j \mid g_j(x) = 0\} \cup \{r\} \tag{8}$$

Observe that if the $k$th region classifier correctly classifies the example $x_i$, i.e., $f_k(x_i) = y_i$ then this would encourage $g_k(x_i) = 0$. This is because $g_k(x_i) = 1$ would induce an increased cost in terms of increasing $L_{k+1}(x_i, y_i)$. Similarly, if the $k$th region classifier incorrectly classifies, namely, $f_k(x_i) \neq y_i$, the optimization would prefer $g_k(x_i) = 1$. Also note that if the kth region classifier loss as well as the subsequent stages are incorrect on an example are incorrect then the weight on that example is zero. This is not surprising since reject/no-reject does not impact the global cost. We can deal with minimizing indicator losses and resulting convergence issues by deriving a global surrogate as we did in Sec. 2.2. A pseudo-code for the proposed scheme is described in Algorithm 1.

## 2.4 Local Linear Classification

Linear classification is a natural method for learning local decision boundaries, with the global decision regions approximated by piecewise linear functions. In local linear classification, local classifiers, $f_1, f_2, \ldots, f_r$, and reject classifiers, $g_1, g_2, \ldots, g_{r-1}$, are optimized over the set of linear functions. Local linear rules can effectively tradeoff bias and variance error. Bias error (empirical error) can be made arbitrarily small by approximating the decision boundary by many local linear classifiers. Variance error (classifier complexity) can be made small by restricting the number of local linear classifiers used to construct the global classifier. This idea is based on the relatively small VC-dimension of a binary local linear classifier, namely,

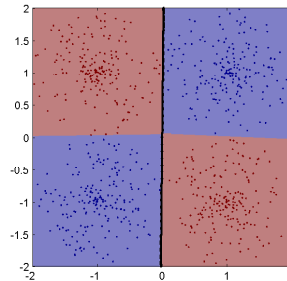

Figure 2: Local LDA classification regions for XOR data, the black line is reject classifier boundary.

**Theorem 2.4.** *The VC-dimension of the class composed (Eq. 8) with $r - 1$ linear classifiers $g_j$ and $r$ linear classifiers $f_j$ in a $d$-dimensional space is bounded by $2(2r - 1)\log(e(2r - 1))(d + 1)$.*

The VC-dimension of local linear classifiers grows linearly with dimension and nearly linearly with respect to the number of regions. This is seen from Fig. 1. In practice, few regions are necessary to achieve low training error as highly non-linear decision boundaries can be approximated well locally with linear boundaries. For example, consider 2-D XOR data. Learning the local linear classifier with 2 regions using LDA produces a classifier with small empirical error. In fact our empirical observation can be translated to a theorem (see Supplementary for details):

**Theorem 2.5.** *Consider an idealized XOR, namely, samples are concentrated into four equal clusters at coordinates* $(-1, 1), (1, 1), (1, -1), (-1, -1)$ *in a 2D space. Then with high probability (where probability is wrt initial sampling of reject region) a two region composite classifier trained locally using LDA converges to zero training error.*

In general, training linear classifiers on the indicator loss is impractical. Optimization on the non-convex problem is difficult and usually leads to non-unique optimal solutions. Although margin based methods such as SVMs can be used, we primarily use relatively simple schemes such as LDA, logistic regression, and average voted perceptron in our experiments. We use each of these schemes for learning both reject and region-classifiers. These schemes enjoy significant computational advantages over other schemes.

**Computational Costs of LDA, Logistic Regression and Perceptron:** Each LDA classifier is trained in $O(nd^2)$ computations, where $n$ is the number of training observations and $d$ is the dimension of the training data. As a result, the total computation cost per iteration of the local linear classifier with LDA scales linearly with respect to the number of training samples, requiring $O(nd^2r)$ computations per iteration, where $r$ is the number of classification regions. Similarly, the computational cost of training a single linear classifier by logistic regression scales $O(ncd^2)$ for a fixed number of iterations, with the local linear classifier training time scaling $O(rncd^2)$ computations per iteration, where $c$ is the number of classes. A linear variant of the voted perceptron was implemented by taking the average of the weights generated by the unnormalized voted perceptron [15]. Training each perceptron for a fixed number of epochs is extremely efficient, requiring only $O(ndc)$ computations to train. Therefore, training local linear perceptron scales linearly with data size and dimensions, with $O(ndcr)$ computations, per iteration.

# 3 Experimental Results

**Multiclass Classification:** Experimental results on six datasets from the UCI repository [16] were performed using the benchmark training and test splits associated with each data set, as shown in Table 1. Confidence intervals are not possible with the results, as the predefined training and test splits were used. Although confidence intervals cannot be computed by multiple training/test splits, test set error bounds [17] show that with test data sets of these sizes, the difference between true error and empirical error is small with high probability. The six datasets tested were: Isolet (d=617, c= 26, n=6238, T=1559), Landsat (d=36, c=7, n=4435, T=2000), Letter (d=16, c=26, n=16000, T=4000), Optdigit (d=64, c=10, n=3823, T=1797), Pendigit (d=16, n=10, n=7494, T=3498), and Shuttle (d=9, c=7, n=43500, T=14500), where $d$ is the dimensions, $c$ the number of classes, $n$ training data size and $T$ the number of test samples.

Local linear classifiers were trained with LDA, logistic regression, and perceptron (mean of weights) used to learn local surrogates for the rejection and local classification problems. The classifiers were initialized with 5 classification regions ($r = 5$), with the trained classifiers often reducing to fewer classification regions due to empty rejection region. Termination of the algorithm occurred when the rejection outputs, $g_k(x)$, and classification labels, $F(x)$, remained consistent on the training data for two iterations. Each classifier was randomly initialized 15 times, and the classifier with the minimum training error was chosen. Results were compared with Mixture Discriminant Analysis (MDA)

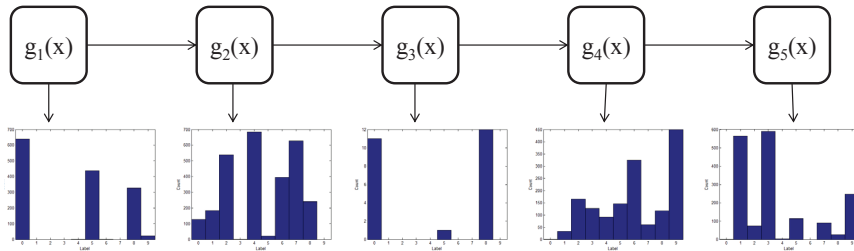

Figure 3: Histogram of classes over test data for the Optdigit dataset in different partitions generated by our approach using the linear voted perceptron .

[9] and classification trees trained using the Gini diversity index (GDI) [3]. These classification algorithms were chosen for comparison as both train global classifiers modeled as simple local classifiers, and both are computationally efficient.

For comparison to globally complex classification techniques, previous state of the art boosting results of Saberian and Vasconcelos [18] and Jhu et al. [19] were listed. Although the multiclass boosted classifiers were terminated early, we consider the comparison appropriate, as early termination limits the complexity of the classifiers. The improved performance of local linear learning of comparable complexity justifies approximating these boundaries by piecewise linear functions. Comparison with kernelized SVM was omitted, as SVM is rarely applied to multiclass learning on large datasets. Training each binary kernelized classifier is computationally intensive, and on weakly learnable data, boosting also allows for modeling of complex boundaries with arbitrarily small empirical error.

Table 1: Multiclass learning algorithm test errors on six UCI datasets using benchmark training and test sets. Bold indicates best test error among listed algorithms. One vs All AdaBoostis trained using decision stumps as weak learners. AdaBoost-SAMME and GD-MCBoost are trained using depth-2 decision trees as weak learners.

| Algorithm | Isolet | Landsat | Letter | Optdigit | Pendigit | Shuttle |
|---|---|---|---|---|---|---|
| One vs All AdaBoost [2] | 11.10% | 16.10% | 37.37% | 12.24% | 11.29% | 0.11% |
| GDI Tree [3] | 20.59% | 14.45% | 14.37% | 14.58% | 8.78% | **0.04**% |
| MDA [9] | 35.98% | 36.45% | 22.73% | 9.79% | 7.75% | 9.59% |
| AdaBoost-SAMME [19] | 39.00% | 20.20% | 44.35% | 22.47% | 16.18% | 0.30% |
| GD-MCBoost [18] | 15.72% | **13.35**% | 40.35% | 7.68% | 7.06% | 0.27% |
| **Local Classifiers** | | | | | | |
| LDA | **5.58**% | 13.95% | 24.45% | 5.78% | 6.60% | 2.67% |
| Logistic Regression | 19.95% | 14.00% | **13.08**% | 7.74% | 4.75% | 1.19% |
| Perceptron | 5.71% | 20.15% | 20.40% | **4.23**% | **4.32**% | 0.32% |

In 4 of the 6 datasets, local linear classification produced the lowest classification error on test datasets, with optimal test errors within 0.6% of the minimal test error methods for the remaining two datasets. Also there is evidence that suggests that our scheme partitions multiclass problems into simpler subproblems. We plotted histogram output of class labels for Optdigit dataset across different regions using local perceptrons (Fig. 3). The histogram is not uniform across regions, implying that the reject classifiers partition easily distinguishable classes. We may interpret our approach as implicitly learning data-dependent codes for multiclass problems. This can contrasted with many state of the art boosting techniques, such as [18], which attempt to optimize both the codewords for each class as well as the binary classification problems defining the codewords.

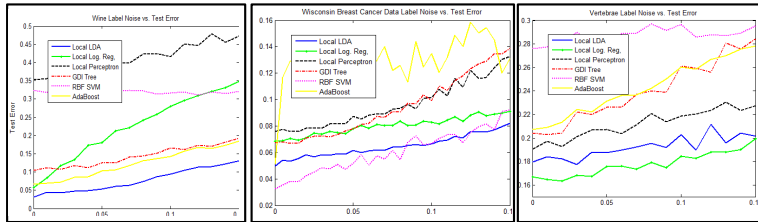

Figure 4: Test error for different values of label noise. **Left:** Wisconsin Breast Cancer data, **Middle:** Vertebrae data, and **Right:** Wine data.

**Robustness to Label Noise:** Local linear classification trained using LDA, logistic regression, and averaged voted perceptron was tested in the presence of random label noise. A randomly selected fraction of all training observations were given incorrect labels, and trained as described for the multiclass experiments. Three datasets were chosen from the UCI repository [16]: Wisconsin Breast Cancer data, Vertebrae data, and Wine data. A training set of 100 randomly selected observations was used, with the remainder of the data used as test. For each label noise fraction, 100 randomly drawn training and test sets were used, and the average test error is shown in Fig. 4.

For comparison, results are shown for classification trees trained according to Gini's diversity index (GDI) [3], AdaBoost trained with stumps [2], and support vector machines trained on Gaussian radial basis function kernels. Local linear classification, notably when trained using LDA, is extremely robust to label noise. In comparison, boosting and classification trees show sensitivity to label noise, with the test error increasing at a faster rate than LDA-trained local linear classification on both the Wisconsin Breast Cancer data and Vertebrae data.

## Acknowledgments

This research was partially supported by NSF Grant 0932114.

# References

[1] G. Rätsch, T. Onoda, and K.-R. Müller. Soft margins for AdaBoost. Technical Report NC-TR-1998-021, Department of Computer Science, Royal Holloway, University of London, Egham, UK, August 1998. Submitted to Machine Learning.

[2] Yoav Freund and Robert E Schapire. A decision-theoretic generalization of on-line learning and an application to boosting. *Journal of Computer and System Sciences*, 55(1):119 – 139, 1997.

[3] Leo Breiman, J. H. Friedman, R. A. Olshen, and C. J. Stone. *Classification and Regression Trees*. Wadsworth, 1984.

[4] Thomas G. Dietterich and Ghulum Bakiri. Solving multiclass learning problems via error-correcting output codes. *Journal of Artificial Intelligence Research*, 2:263–286, 1995.

[5] Erin L. Allwein, Robert E. Schapire, and Yoram Singer. Reducing multiclass to binary: a unifying approach for margin classifiers. *J. Mach. Learn. Res.*, 1:113–141, September 2001.

[6] Koby Crammer and Yoram Singer. On the learnability and design of output codes for multiclass problems. In *In Proceedings of the Thirteenth Annual Conference on Computational Learning Theory*, pages 35–46, 2000.

[7] Venkatesan Guruswami and Amit Sahai. Multiclass learning, boosting, and error-correcting codes. In *Proceedings of the twelfth annual conference on Computational learning theory*, COLT '99, pages 145–155, New York, NY, USA, 1999. ACM.

[8] Yijun Sun, Sinisa Todorovic, Jian Li, and Dapeng Wu. Unifying the error-correcting and output-code adaboost within the margin framework. In *Proceedings of the 22nd international conference on Machine learning*, ICML '05, pages 872–879, New York, NY, USA, 2005. ACM.

[9] Trevor Hastie and Robert Tibshirani. Discriminant analysis by gaussian mixtures. *Journal of the Royal Statistical Society, Series B*, 58:155–176, 1996.

[10] Tae-Kyun Kim and Josef Kittler. Locally linear discriminant analysis for multimodally distributed classes for face recognition with a single model image. *IEEE Transactions on Pattern Analysis and Machine Intelligence*, 27:318–327, 2005.

[11] Ofer Dekel and Ohad Shamir. There's a hole in my data space: Piecewise predictors for heterogeneous learning problems. In *Proceedings of the International Conference on Artificial Intelligence and Statistics*, volume 15, 2012.

[12] Juan Dai, Shuicheng Yan, Xiaoou Tang, and James T. Kwok. Locally adaptive classification piloted by uncertainty. In *Proceedings of the 23rd international conference on Machine learning*, ICML '06, pages 225–232, New York, NY, USA, 2006. ACM.

[13] Marc Toussaint and Sethu Vijayakumar. Learning discontinuities with products-of-sigmoids for switching between local models. In *Proceedings of the 22nd international conference on Machine Learning*, pages 904–911. ACM Press, 2005.

[14] Eduardo D. Sontag. Vc dimension of neural networks. In *Neural Networks and Machine Learning*, pages 69–95. Springer, 1998.

[15] Yoav Freund and Robert E. Schapire. Large margin classification using the perceptron algorithm. *Machine Learning*, 37:277–296, 1999. 10.1023/A:1007662407062.

[16] A. Frank and A. Asuncion. UCI machine learning repository, 2010.

[17] J. Langford. Tutorial on practical prediction theory for classification. *Journal of Machine Learning Research*, 6(1):273, 2006.

[18] Mohammad J. Saberian and Nuno Vasconcelos. Multiclass boosting: Theory and algorithms. In J. Shawe-Taylor, R.S. Zemel, P. Bartlett, F.C.N. Pereira, and K.Q. Weinberger, editors, *Advances in Neural Information Processing Systems 24*, pages 2124–2132. 2011.

[19] Ji Zhu, Hui Zou, Saharon Rosset, and Trevor Hastie. Multi-class adaboost, 2009.

